# Feature Set Embedding for Incomplete Data

**David Grangier**
NEC Labs America
Princeton, NJ
dgrangier@nec-labs.com

**Iain Melvin**
NEC Labs America
Princeton, NJ
iain@nec-labs.com

## Abstract

We present a new learning strategy for classification problems in which train and/or
test data suffer from missing features. In previous work, instances are represented
as vectors from some feature space and one is forced to impute missing values or
to consider an instance-specific subspace. In contrast, our method considers in-
stances as sets of (feature,value) pairs which naturally handle the missing value
case. Building onto this framework, we propose a classification strategy for sets.
Our proposal maps (feature,value) pairs into an embedding space and then non-
linearly combines the set of embedded vectors. The embedding and the combina-
tion parameters are learned jointly on the final classification objective. This simple
strategy allows great flexibility in encoding prior knowledge about the features in
the embedding step and yields advantageous results compared to alternative solu-
tions over several datasets.

## 1 Introduction

Many applications require classification techniques dealing with train and/or test instances with miss-
ing features: e.g. a churn predictor might deal with incomplete log features for new customers,
a spam filter might be trained from data originating from servers storing different features, a face
detector might deal with images for which high resolution cues are corrupted.

In this work, we address a learning setting in which the missing features are either missing at ran-
dom [6], i.e. deletion due to corruption or noise, or structurally missing [4], i.e. some features do not
make sense for some examples, e.g. activity history for new customers. We do not consider setups
in which the features are maliciously deleted to fool the classifier [5]. Techniques for dealing with
incomplete data fall mainly into two categories: techniques which impute the missing features and
techniques considering an instance-specific subspace.

Imputation-based techniques are the most common. In this case, the data instances are viewed as
feature vectors in a high-dimensional space and the classifier is a function from this space into the
discrete set of classes. Prior to classification, the missing vector components need to be imputed.
Early imputation approaches fill any missing value with a constant, zero or the average of the feature
over the observed cases [18]. This strategy neglects inter-feature correlation, and completion tech-
niques based on k-nearest-neighbors (k-NN) have subsequently been proposed to circumvent this
limitation [1]. Along this line, more complex strategies based on generative models have been used
to fill missing features according to the most likely value given the observed features. In this case, the
Expectation-Maximization algorithm is typically adopted to estimate the data distribution over the
incomplete training data [9]. Building upon this generative model strategy, several approaches have
considered integrating out the missing values, either by integrating the loss [2] or the decision func-
tion [22]. Recently, [15] and [6] have proposed to avoid the initial maximum likelihood distribution
estimation. Instead, they proposed to learn jointly the generative model and the decision function to
optimize the final classification loss.

As an alternative to imputation-based approaches, [4] has proposed a different framework. In this
case, each instance is viewed as a vector from a subspace of the feature space determined by its

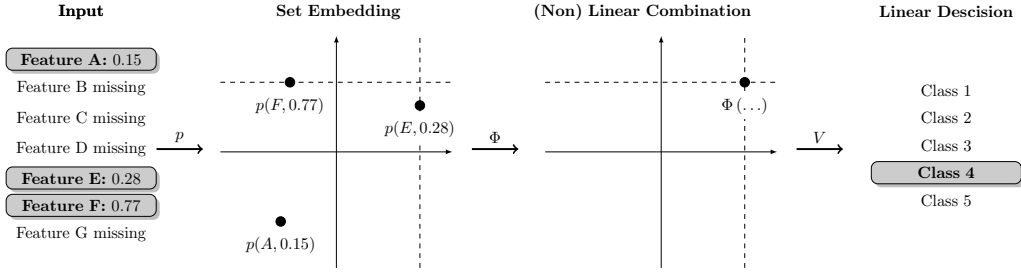

Figure 1: Feature Set Embedding: An example is given a set of (feature, value) pairs. Each pair is mapped into an embedding space, then the embedded vectors are combined into a single vector (either linearly with mean or non-linearly with max). Linear classification is then applied. Our learning procedure learns both the embedding space and the linear classifier jointly.

observed features. A decision function is learned for each specific subspace and parameter sharing between the functions allows the method to achieve tractability and generalization. Compared to imputation-based approaches, this strategy avoids choosing a generative model, i.e. making an assumption about the missing data. Other alternatives to imputation have been proposed in [10] and [5]. These approaches focus on linear classifiers and propose learning procedures which avoid concentrating the weights on a small subset of the features, which helps achieve better robustness with respect to feature deletion.

In this work, we propose a novel strategy called *feature set embedding*. Contrary to previous work, we do not consider instances as vectors from a given feature space. Instead, we consider instances as a set of (feature, value) pairs and propose to learn to classify sets directly. For that purpose, we introduce a model which maps each (feature, value) pair onto an embedding space and combines the embedded pairs into a single vector before applying a linear classifier, see Figure 1. The embedding space mapping and the linear classifier are jointly learned to maximize the conditional probability of the label given the observed input. Contrary to previous work, this set embedding framework naturally handles incomplete data without modeling the missing feature distribution, or considering an instance specific decision function. Compared to other work on learning from sets, our approach is original as it proposes to learn to embed set elements and to classify sets as a single optimization problem, while prior strategies learn their decision function considering a fixed mapping from sets into a feature space [12, 3].

The rest of the paper is organized as follows: Section 2 presents the proposed approach, Section 3 describes our experiments and results. Section 4 concludes.

## 2 Feature Set Embedding

We denote an example as $(X, y)$ where $X = \{(f_i, v_i)\}_{i=1}^{|X|}$ is a set of (feature, value) pairs and $y$ is a class label in $\mathcal{Y} = \{1, \ldots, k\}$. The set of features is discrete, i.e. $\forall i, f_i \in \{1, \ldots d\}$, while the feature values are either continuous or discrete, i.e. $\forall i, v_i \in \mathcal{V}_{f_i}$ where $\mathcal{V}_{f_i} = \mathbb{R}$ or $\mathcal{V}_{f_i} = \{1, \ldots, c_{f_i}\}$. Given a labeled training dataset $D_{\text{train}} = \{(X_i, y_i)\}_{i=1}^{n}$, we propose to learn a classifier $g$ which predicts a class from an input set $X$.

For that purpose, we combine two levels of modeling. At the lower level, (feature, value) pairs are individually mapped into an embedding space of dimension $m$: given an example $X = \{(f_i, v_i)\}_{i=1}^{|X|}$, a function $p$ predicts an embedding vector $p_i = p(f_i, v_i) \in \mathbb{R}^m$ for each feature value pair $(f_i, v_i)$. At the upper level, the embedded vectors are combined to make the class prediction: a function $h$ takes the set of embedded vectors $\{p_i\}_{i=1}^{|X|}$ and predicts a vector of confidence values $h(\{p_i\}_{i=1}^{|X|}) \in \mathbb{R}^k$ in which the correct class should be assigned the highest value. Our classifier composes the two levels, i.e $g = h \circ p$. Intuitively, the first level extracts the information relevant to class prediction provided by each feature, while the second level combines this information over all observed features.

## 2.1 Feature Embedding

Feature embedding offers great flexibility. It can accommodate discrete and continuous data and allows encoding prior knowledge on characteristics shared between groups of features. For discrete features, the simplest embedding strategy learns a distinct parameter vector for each $(f, v)$ pair, i.e.

$$p(f, v) = L_{f,v} \text{ where } L_{f,v} \in \mathbb{R}^m.$$

For capacity control, rank regularization can be applied,

$$p(f, v) = W L_{f,v} \text{ where } L_{f,v} \in \mathbb{R}^l \text{ and } W \in \mathbb{R}^{m \times l},$$

In this case, $l < m$ is a hyperparameter bounding the rank of $WL$, where $L$ denotes the matrix concatenating all $L_{f,v}$ vectors. One can further indicate that two pairs $(f, v)$ and $(f, v')$ originate from the same feature by parameterizing $L_{f,v}$ as

$$L_{f,v} = \begin{bmatrix} L_f^{(a)} \\ L_{f,v}^{(b)} \end{bmatrix} \text{ where } \begin{cases} L_f^{(a)} \in \mathbb{R}^{l^{(a)}} \text{ and } L_{f,v}^{(b)} \in \mathbb{R}^{l^{(b)}} \\ l^{(a)} + l^{(b)} = l \end{cases} \tag{1}$$

Similarly, one can indicate that two pairs $(f, v)$ and $(f', v)$ shares the same value by parameterizing,

$$L_{f,v} = \begin{bmatrix} L_{f,v}^{(a)} \\ L_v^{(b)} \end{bmatrix} \text{ where } \begin{cases} L_{f,v}^{(a)} \in \mathbb{R}^{l^{(a)}} \text{ and } L_v^{(b)} \in \mathbb{R}^{l^{(b)}} \\ l^{(a)} + l^{(b)} = l \end{cases} \tag{2}$$

This is useful when feature values share a common physical meaning, like gray levels at different pixel locations or temperatures measured by different sensors. Of course, the parameter sharing strategies (1) and (2) can be combined.

When the feature values are continuous, we adopt a similar strategy and define

$$p(f, v) = W \begin{bmatrix} L_f^{(a)} \\ v L_f^{(b)} \end{bmatrix} \text{ where } \begin{cases} L_f^{(a)} \in \mathbb{R}^{l^{(a)}} \text{ and } L_f^{(b)} \in \mathbb{R}^{l^{(b)}} \\ l^{(a)} + l^{(b)} = l \end{cases} \tag{3}$$

where $L_f^{(a)}$ informs about the presence of feature $f$, while $v L_f^{(b)}$ informs about its value. If the model is thought not to need presence information, $L_f^{(a)}$ can be omitted, i.e. $l^{(a)} = 0$.

When the dataset contains a mix of continuous and discrete features, both embedding approaches can be used jointly. Feature embedding is hence a versatile strategy; the practitioner defines the model parameterization according to the nature of the features, and the learned parameters $L$ and $W$ encode the correlation between features.

## 2.2 Classifying from an Embedded Feature Set

The second level of our architecture $h$ considers the set of embedded features and predicts a vector of confidence values. Given an example $X = \{(f_i, v_i)\}_{i=1}^{|X|}$, the function $h$ takes the set $P = \{p(f_i, v_i)\}_{i=1}^{|X|}$ as input, and outputs $h(P) \in \mathbb{R}^k$ according to

$$h(P) = V \Phi(P)$$

where $\Phi$ is a function which takes a set of vector of $\mathbb{R}^m$ and outputs a single vector of $\mathbb{R}^m$, while $V$ is a $k$-by-$m$ matrix. This second level is hence related to kernel methods for sets, which first apply a fixed mapping $\Phi$ from sets to vectors, before learning a linear classifier in the feature space [12]. In our case, however, we make sure that $\Phi$ is a generalized differentiable function [19], so that $h$ and $p$ can be optimized jointly. In the following, we consider two alternatives for $\Phi$: a linear function, the mean, and a non-linear function, the component-wise max.

**Linear Model**    In this case, one can remark that

$$\begin{aligned} h(P) &= V \text{ mean}(\{p(f_i, v_i)\}_{i=1}^{|X|}) \\ &= V \text{ mean}(\{W L_{f_i, v_i}\}_{i=1}^{|X|}) \\ &= V W \text{ mean}(\{L_{f_i, v_i}\}_{i=1}^{|X|}) \end{aligned}$$

by linearity of the mean. Hence, in this case, the dimension of the embedding space $m$ bounds the rank of the matrix $VW$. This also means that considering $m > k$ is irrelevant in the linear case. In the specific case where features are continuous and no presence information is provided, i.e. $L_{f,v} = vL_f^{(b)}$, our model is equivalent to a classical linear classifier operating on feature vectors when all features are present, i.e. $|X| = d$,

$$g(X) = VW \, \text{mean}(\{L_{f_i, v_i}\}_{i=1}^d) = \frac{1}{d} VW \sum_{i=1}^d v_i L_{f_i}^{(b)} = \frac{1}{d}(VWL)v$$

where $L$ denotes the matrix $[L_{f_1}^{(b)}, \ldots, L_{f_d}^{(b)}]$ and $v$ denotes the vector $[v_1, \ldots, v_d]$. Hence, in this case, our model corresponds to

$$g(X) = Mv \text{ where } M \in \mathbb{R}^{k \times d} \text{ s.t. rank(M)} = \min\{k, l, m, d\}$$

**Non-linear Model**  In this case, we rely on the component-wise max. This strategy can model more complex decision functions. In this case, selecting $m > k, l$ is meaningful. Intuitively, each dimension in the embedding space provides a meta-feature describing each (feature, value) pair, the max operator then outputs the best meta-feature match over the set of (feature, value) pairs, performing a kind of soft-OR, i.e. checking whether there is at least one pair for which the meta-feature is high. The final classification decision is then taken as a linear combination of the $m$ soft-ORs. One can relate our use of the max operator to its common use in fixed set mapping for computer vision [3].

## 2.3 Model Training

Model learning aims at selecting the parameter matrices $L$, $W$ and $V$. For that purpose, we maximize the (log) posterior probability of the correct class over the training set $D_{\text{train}} = \{(X_i, y_i)\}_{i=1}^n$, i.e.

$$C = \sum_{i=1}^n \log P(y_i | X_i)$$

where model outputs are mapped to probabilities through a softmax function, i.e.

$$P(y|X) = \frac{\exp(g(X)_y)}{\sum_{y'=1}^k \exp(g(X)_{y'})} \,.$$

Capacity control is achieved by selecting the hyperparameters $l$ and $m$. For linear models, the criterion $C$ is referred to as the multiclass logistic regression objective and [16] has studied the relation between $C$ and margin maximization. In the binary case ($k = 2$), the criterion $C$ is often referred to as the *cross entropy* objective.

The maximization of $C$ is conducted through stochastic gradient ascent for random initial parameters. This algorithm enables the addressing of large training sets and has good properties for non-convex problems [14], which is of interest for our non-linear model and for the linear model when rank regularization is used. One can note that our non-linear model relies on the max function, which is not differentiable everywhere. However, [8] has shown that gradient ascent can also be applied to generalized differentiable functions, which is the case of our criterion.

## 3 Experiments

Our experiments consider different setups: features missing at train and test time, features missing only at train time, features missing only at test time. In each case, our model is compared to alternative solutions relying on experimental setups introduced in prior work. Finally, we study our model in various conditions over the larger MNIST dataset.

### 3.1 Missing Features at Train and Test Time

The setup in which features are missing at train and test time is relevant to applications suffering sensor failure or communication errors. It is also relevant to applications in which some features are

Table 1: Dataset Statistics

|  | Train set size | Test set size | # eval. splits | Total # feat. | Missing feat.(%) | Continuous or discrete |
|---|---|---|---|---|---|---|
| UCI sick | 2,530 | 633 | 5 | 25 | 90 | c |
| pima | 614 | 154 | 5 | 8 | 90 | c |
| hepatitis | 124 | 31 | 5 | 19 | 90 | c |
| echo | 104 | 27 | 5 | 7 | 90 | c |
| hypo | 2,530 | 633 | 5 | 25 | 90 | c |
| MNIST-5-vs-6 | 1,000 | 200 | 2 | 784 | 25 | d |
| Cars | 177 | 45 | 5 | 1,900 | 62 | d |
| USPS | 1,000 | 6,291 | 100 | 256 | 85* | c |
| Physics | 1,000 | 5,179 | 100 | 78 | 85* | c |
| Mine | 500 | 213 | 100 | 41 | 26* | c |
| MNIST-miss-test[†] | 12×100 | 12×300 | 20 | 784 | 0 to 99[†] | d |
| MNIST-full | 60,000 | 10,000 | 1 | 784 | 0 to 87 | d |

\* Features missing only at training time for USPS, Physics and Mine.

[†] Features missing only at test time for MNIST-miss-test. This set presents 12 binary problems, 4vs9, 3vs5, 7vs9, 5vs8, 3vs8, 2vs8, 2vs3, 8vs9, 5vs6, 2vs7, 4vs7 and 2vs6, each having 100 examples for training, 200 for validation and 300 for test.

structurally missing, i.e. the measurements are absent because they do not make sense (e.g. see the car detection experiments).

We compare our model to alternative solutions over the experimental setup introduced in [4]. Three sets of experiments are considered. The first set relies on binary classification problems from the UCI repository. For each dataset, $90\%$ of the features are removed at random. The second set of experiments considers the task of discriminating between handwritten characters of 5 and 6 from the MNIST dataset. Contrary to UCI, the deleted features have some structure; for each example, a square area covering $25\%$ of the image surface is removed at random. The third set of experiments considers detecting cars in images. This task presents a problem where some features are structurally missing. For each example, regions of interests corresponding to potential car parts are detected, and features are extracted for each region. For each image, 19 types of region are considered and between 0 and 10 instances of each region have been extracted. Each region is then described by 10 features. This region extraction process is described in [7]. Hence, at most $1900 = 19 \times 10 \times 10$ features are provided for each image. Data statistics are summarized in Table 1.

On these datasets, *Feature Set Embedding (FSE)* is compared to 7 baseline models. These baselines are all variants of Support Vector Machines (SVMs), suitable for the missing feature problem. *Zero*, *Mean*, *GMM* and *kNN* are imputation-based strategies: *Zero* sets the missing values to zero, *Mean* sets the missing values to the average value of the features over the training set, *GMM* finds the most likely missing values given the observed ones relying on a Gaussian Mixture learned over the training set, *kNN* fills the missing values of an instance based on its k-nearest-neighbors, relying on the Euclidean distance in the subspace relevant to each pair of examples. *Flag* relies on the *Zero* imputation but complements the examples with binary features indicating whether each feature was observed or imputed. Finally, *geom* is a subspace-based strategy [4]; for each example, a classifier in the subspace corresponding to the observed features is considered. The instance-specific margin is maximized but the instance-specific classifiers share common weights.

For each experiment, the hyperparameters of our model $l$, $m$ and the number of training iterations are validated by first training the model on $4/5$ of the training data and assessing it on the remainder of the training data. A similar strategy has been used for selecting the baseline parameters. The SVM kernel has notably been validated between linear and polynomial up to order $3$. Test performance is then reported over the best validated parameters.

Table 2 reports the results of our experiments. Overall, FSE performs at least as well as the best alternative for all experiments, except for hepatitis where all models yield almost the same performance. In the case of structurally missing features, the car experiment shows a substantial advantage for FSE over the second best approach *geom*, which was specifically introduced for this kind of setup. During validation (no validation results are reported due to space constraints), we noted that non-linear mod-

Table 2: Error Rate (%) for Missing Features at Train & Test Time

|  |  | FSE | geom | zero | mean | flag | GMM | kNN |
|---|---|---|---|---|---|---|---|---|
| UCI | sick | 9 | 10 | 9 | 37 | 16 | 40 | 30 |
|  | pima | 34 | 34 | 34 | 35 | 35 | 35 | 41 |
|  | hepatitis | 23 | 22 | 22 | 22 | 22 | 22 | 23 |
|  | echo | 33 | 34 | 37 | 33 | 36 | 33 | 33 |
|  | hypo | 5 | 5 | 7 | 35 | 6 | 33 | 19 |
| MNIST-5-vs-6 |  | 5 | 5 | 5 | 6 | 7 | 5 | 6 |
| Cars |  | 24 | 28 | 39 | 39 | 41 | 38 | 48 |

Table 3: Error rate (%) for missing features at train time only

|  | FSE | meanInput | GMM | meanFeat |
|---|---|---|---|---|
| USPS | 11.7 | 13.6 | 9.0 | 13.2 |
| Physics | 23.8 | 29.2 | 31.2 | 29.6 |
| Mines | 9.8 | 11.7 | 10.5 | 10.6 |

els, i.e. the baseline SVM with a polynomial kernel of order 2 and FSE with $\phi = \max$, outperformed their linear counterparts. We therefore solely validate non-linear FSE in the following: For feature embedding of continuous data, feature presence information has proven to be useful in all cases, i.e. $l^{(a)} > 0$ in Eq. (3). For feature embedding of discrete data, sharing parameters across different values of the same feature, i.e. Eq. (1), was also helpful in all cases. We also relied on sharing parameters across different features with the same value, i.e. Eq. (2), for datasets where the feature values shared a common meaning, i.e. gray levels for MNIST and region features for cars. For the hyperparameters $(l, m)$ of our model, we observed that the main control on our model capacity is the embedding size $m$. Its selection is simple since varying this parameter consistently yields convex validation curves. The rank regularizer $l$ needed little tuning, yielding stable validation performance for a wide range of values.

### 3.2 Missing Features at Train Time

The setup presenting missing features at training time is relevant to applications which rely on different sources for training. Each source might not collect the exact same set of features, or might have introduced novel features during the data collection process. At test time however, the feature detector can be designed to collect the complete feature set.

In this case, we compare our model to alternative solutions over the experimental setup introduced in [6]. Three datasets are considered. The first set USPS considers the task of discriminating between odd and even handwritten digits over the USPS dataset. The training set is degraded and $85\%$ of the features are missing. The second set considers the quantum physics data from the KDD Cup 2004 in which two types of particles generated in high energy collider experiments should be distinguished. Again, the training set is degraded and $85\%$ of the features are missing. The third set considers the problem of detecting land-mines from 4 types of sensors, each sensor provides a different set of features or views. In this case, for each instance, whole views are considered missing during training. Data statistics are summarized in Table 1 for the three sets.

For this set of experiments, we rely on *infinite imputations* as a baseline. Infinite imputation is a general technique proposed for the case where features are missing at train time. Instead of pretraining the distribution governing the missing values with a generative objective, infinite imputations proposes to train the imputation model and the final classifier in a joint optimization framework [6]. In this context, we consider an SVM with a RBF kernel as the classifier and three alternative imputation models *Mean*, *GMM* and *MeanFeat* which corresponds to mean imputations in the feature space. For each experiment, we follow the validation strategy defined in the previous section for FSE. The validation strategy for tuning the parameters of the other models is described in [6].

Table 3 reports our results. FSE is the best model for the Physics and Mines dataset, and the second best model for the USPS dataset. In this case, features are highly correlated and *GMM* imputation yields a challenging baseline. On the other hand, Physics presents a challenging problem with higher

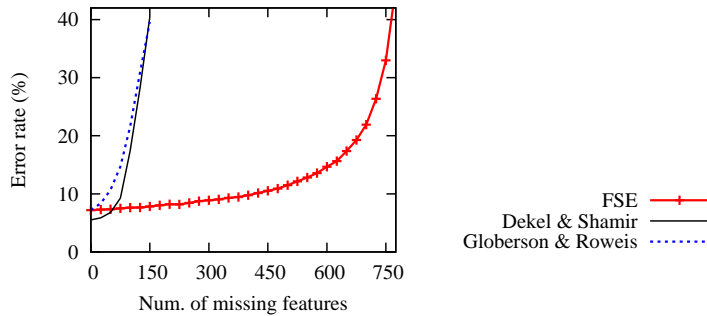

Figure 2: Results for MNIST-miss-test (12 binary problems with features missing at test time only)

error rates for all models. In this case, feature correlation is low and *GMM* imputation is yielding the worse performance, while our model brings a strong improvement.

### 3.3 Missing Features at Test Time

The setup presenting missing features at test time considers applications in which the training data have been produced with more care than the test data. For example, in a face identification application, customers could provide clean photographs for training while, at test time, the system should be required to work in the presence of occlusions or saturated pixels.

In this case, we compare our work to [10] and [5]. Both strategies propose to learn a classifier which avoids assigning high weight to a small subset of features, hence limiting the impact of the deletion of some features at test time. [10] formulates their strategy as a min-max problem, i.e. identifying the best classifier under the worst deletion, while [5] relies on an $L_\infty$ regularizer to avoid assigning high weights to few features. We compare our algorithm to these alternatives over binary problems discriminating handwritten digits originating from MNIST. This experimental setup has been introduced in [10] and Table 1 summarizes its statistics. In this setup, the data is split into training, validation and test sets. For a fair comparison, the validation set is used solely to select hyperparameters, i.e. we do not retrain the model over both training and validation sets after hyperparameter selection.

Since no features are missing at train time, we adapt our training procedure to take into account the mismatched conditions between train and test. Each time an example is considered during our stochastic training procedure, we delete a random subset of its features. The size of this subset is sampled uniformly between 0 and the total number of features minus 1.

Figure 2 plots the error rate as a function of the number of missing features. FSE has a clear advantage in most settings: it achieves a lower error rate than Globerson & Roweis [10] in all cases, while it is better than Dekel & Shamir [5], as soon as the number of missing features is above 50, i.e. less than 6% missing features. In fact, we observe that FSE is very robust to feature deletion; its error rate remains below 20% for up to 700 missing features i.e. 90% missing features. On the other end, the alternative strategies report performance close to random when the number of missing features reaches 150, i.e. 20% missing features. Note that [10] and [5] further evaluate their models in an adversarial setting, i.e. features are intentionally deleted to fool the classifier, that is beyond the scope of this work.

### 3.4 MNIST-full experiments

The previous experiments compared our model to prior approaches relying on the experimental setups introduced to evaluate these approaches. These setups proposed small training sets motivated by the training cost of the compared alternatives (see Table 1). In this section, we stress the scalability of our learning procedure and study FSE on the whole MNIST dataset with 10 classes and 60,000 training instances. All conditions are considered: features missing at training time, at testing time, and at both times.

We train 4 models which have access to training sets with various numbers of available features, i.e. 100, 200, 500 and 784 features which approximately correspond to 90, 60, 35 and 0% missing

Table 4: Error Rate (%) 10-class MNIST-full Experiments

| # train f. | # test features | | | |
|---|---|---|---|---|
| | 100 | 300 | 500 | 784 |
| 100 | 19.8 | 8.9 | 7.5 | 6.9 |
| 300 | 34.2 | 7.4 | 4.8 | 3.9 |
| 500 | 55.6 | 12.3 | 4.8 | 2.9 |
| 784 | 78.3 | 46.7 | 17.8 | 2.5 |
| random | 10.7 | 2.9 | 2.1 | 1.8 |

features. We train a $5^{\text{th}}$ model referred to as *random* with the algorithm introduced in Section 3.3, i.e. all training features are available but the training procedure randomly hides some features each time it examines an example. All models are evaluated with $100, 200, 500$ and $784$ available features.

Table 4 reports the results of these experiments. Excluding the *random* model, the result matrix is strongly diagonal, e.g. when facing a test problem with $300$ available features, the model trained with $300$ features is better than the models trained with $100, 500$ or $784$ features. This is not surprising as the training distribution is closer to the testing distribution in that case. We also observe that models facing less features at test time than at train time yield poor performance, while the models trained with few features yield satisfying performance when facing more features. This seems to suggest that training with missing features yields more robust models as it avoids the decision function to rely solely on few specific features that might be corrupted. In other word, training with missing features seems to achieve a similar goal as $L_\infty$ regularization [5]. This observation is precisely what led us to introduce the *random* training procedure. In this case, the model performs better than all other models in all conditions, even when all features are present, confirming our regularization hypothesis. In fact, the results obtained with no missing features ($1.8\%$ error) are comparable to the best non-convolutional methods, including traditional neural networks ($1.6\%$ error) [20]. Only recent work on Deep Boltzmann Machines [17] achieved significantly better performance ($0.95\%$ error). The regularization effect of missing training features could be related to noise injection techniques for regularization [21, 11].

# 4 Conclusions

This paper introduces *Feature Set Embedding* for the problem of classification with missing features. Our approach deviates from the standard classification paradigm: instead of considering examples as feature vectors, we consider examples as sets of (feature, value) pairs which handle the missing feature problem more naturally. In order to classify sets, we propose a new strategy relying on two levels of modeling. At the first level, each (feature, value) is mapped onto an embedding space. At the second level, the set of embedded vectors is compressed onto a single embedded vector over which linear classification is applied. Our training algorithm then relies on stochastic gradient ascent to jointly learn the embedding space and the final linear decision function.

This proposed strategy has several advantages compared to prior work. First, sets are conceptually better suited than vectors for dealing with missing values. Second, embedding (feature, value) pairs offers a flexible framework which easily allows encoding prior knowledge about the features. Third, our experiments demonstrate the effectiveness and the scalability of our approach.

From a broader perspective, the flexible feature embedding framework could go beyond the missing feature application. In particular, it allows using meta-features (attributes describing a feature) [13], e.g. the embedding vector of the temperature features in a weather prediction system could be computed from the locations of their sensors. It also enables the designing of a system in which new sensors are added without requiring full model re-training; in this case, the model could be quickly adapted by only updating embedding vectors corresponding to the new sensors. Also, our approach of relying on feature sets offers interesting opportunities for feature selection and adversarial feature deletion. We plan to study these aspects in the future.

**Acknowledgments** The authors are grateful to Gal Chechik and Uwe Dick for sharing their data and experimental setups.

# References

[1] G. Batista and M. Monard. A study of k-nearest neighbour as an imputation method. In *Hybrid Intelligent Systems (HIS)*, pages 251–260, 2002.

[2] C. Bhattacharyya, P. K. Shivaswamy, and A. Smola. A second order cone programming formulation for classifying missing data. In *Neural Information Processing Systems (NIPS)*, pages 153–160, 2005.

[3] S. Boughhorbel, J-P. Tarel, and F. Fleuret. Non-mercer kernels for svm object recognition. In *British Machine Vision Conference (BMVC)*, 2004.

[4] G. Chechik, G. Heitz, G. Elidan, P. Abbeel, and D. Koller. Max margin classification of data with absent features. *Journal of Machine Learning Research (JMLR)*, 9:1–21, 2008.

[5] O. Dekel, O. Shamir, and L. Xiao. Learning to classify with missing and corrupted features. *Machine Learning Journal*, 2010 (to appear).

[6] U. Dick, P. Haider, and T. Scheffer. Learning from incomplete data with infinite imputations. In *International Conference on Machine Learning (ICML)*, 2008.

[7] G. Elidan, G. Heitz, and D. Koller. Learning object shape: From drawings to images. In *Conference on Computer Vision and Pattern Recognition (CVPR)*, pages 2064–2071, 2006.

[8] Y. M. Ermoliev and V. I. Norkin. Stochastic generalized gradient method with application to insurance risk management. Technical Report 21, International Institute for Applied Systems Analysis, 1997.

[9] Z. Ghahramani and M. I. Jordan. Supervised learning from incomplete data via an em approach. In *Neural Information Processing Systems (NIPS)*, pages 120–127, 1993.

[10] A. Globerson and S. Roweis. Nightmare at test time: robust learning by feature deletion. In *International Conference on Machine Learning (ICML)*, pages 353–360, 2006.

[11] Y. Grandvalet, S. Canu, and S. Boucheron. Noise injection: Theoretical prospects. *Neural Computation*, 9(5):1093–1108, 1997.

[12] R. Kondor and T. Jebara. A kernel between sets of vectors. In *International Conference on Machine Learning (ICML)*, 2003.

[13] E. Krupka, A. Navot, and N. Tishby. Learning to select features using their properties. *Journal of Machine Learning Research (JMLR)*, 9:2349–2376, 2008.

[14] Y. LeCun, L. Bottou, G. B. Orr, and K. R. Mueller. Efficient backprop. In G. B Orr and K. R. Mueller, editors, *Neural Networks: Tricks of the Trade*, chapter 1, pages 9–50. Springer, 1998.

[15] X. Liao, H. Li, and L. Carin. Quadratically gated mixture of experts for incomplete data classification. In *International Conference on Machine Learning (ICML)*, pages 553–560, 2007.

[16] S. Rosset, J. Zhu, and T. Hastie. Margin maximizing loss functions. In *Neural Information Processing Systems (NIPS)*, 2003.

[17] R. Salakhutdinov and H. Larochelle. Efficient learning of deep Boltzmann machines. In *Artificial Intelligence and Statistics (AISTATS)*, 2010.

[18] J.L. Schafer. *Analysis of Incomplete Multivariate Data*. Chapman & Hall, London, UK, 1998.

[19] N.Z. Shor. *Minimization Methods for Non-Differentiable Functions and Applications*. Springer, Berlin, Germany, 1985.

[20] P. Simard, D. Steinkraus, and J.C. Platt. Best practices for convolutional neural networks applied to visual document analysis. In *International Conference on Document Analysis and Recognition (ICDAR)*, pages 958–962, 2003.

[21] P. Vincent, H. Larochelle, Y. Bengio, and P.A. Manzagol. Extracting and composing robust features with denoising autoencoders. In *International Conference on Machine Learning (ICML)*, pages 1096–1103, 2008.

[22] D. Williams, X. Liao, Y. Xue, and L. Carin. Incomplete-data classification using logistic regression. In *International Conference on Machine Learning (ICML)*, pages 972–979, 2005.

